# A selective attention multi–chip system with dynamic synapses and spiking neurons

**Chiara Bartolozzi**
Institute of neuroinformatics
UNI-ETH Zurich
Wintherthurerstr. 190, 8057, Switzerland
chiara@ini.phys.ethz.ch

**Giacomo Indiveri**
Institute of neuroinformatics
UNI-ETH Zurich
Wintherthurerstr. 190, 8057, Switzerland
giacomo@ini.phys.ethz.ch

## Abstract

Selective attention is the strategy used by biological sensory systems to solve the problem of limited parallel processing capacity: salient subregions of the input stimuli are serially processed, while non–salient regions are suppressed. We present an mixed mode analog/digital Very Large Scale Integration implementation of a building block for a multi–chip neuromorphic hardware model of selective attention. We describe the chip's architecture and its behavior, when its is part of a multi–chip system with a spiking retina as input, and show how it can be used to implement in real-time flexible models of bottom-up attention.

## 1 Introduction

Biological systems interact with the outside world in real-time, reacting to complex stimuli in few milliseconds. This is a highly demanding computational task, that requires either very high speed sequential computation or fast massively parallel processing. Real systems however have to cope with limited resources. Biological systems solve this issue by sequentially allocating computational resources on small regions of the input stimuli, for analyzing them in parallel, with a strategy known as Selective Attention, that takes advantage of both sequential and parallel processing.

A wise approach to the design of artificial systems that need to interact with the real world in real time is to take inspiration from the strategies developed by biological systems.

The psychophysical study of selective attention distinguished two complementary strategies for the selection of salient regions of the input stimuli, one depending on the physical (bottom-up) characteristic of the input, the other depending on its semantic (top-down) and task related properties.

Much of the applied research has focused on modeling the bottom-up aspect of selective attention. As a consequence, several software [1, 2, 3] and hardware models [4, 5, 6, 7] based on the concept of *saliency map*, *winner-takes-all* (WTA) competition, and *inhibition of return* (IOR) [8] have been proposed.

We focus on HW implementation of such selective attention systems on compact, low-power, analogue VLSI chips. Previous implementations focused on either very abstract object/attention WTA architectures with dedicated single-chip solutions [6], or on very detailed models of spike-based competitive networks [9]; we propose a multi-chip solution that combines the advantages of spike-based solutions for communicating signals across chips, with a dedicated and compact WTA architecture for implementing competition among a large number of elements in parallel. Specifically we present a new chip with $32 \times 32$ cells, that can sequentially select the most active regions of the input stimuli, the Selective Attention Chip (SAC). It is a transceiver chip employing a spike-based representation (AER, Address-Event-Representation [10]). Its input signals topographically encode the local conspicuousness of the input over the entire visual scene. Its output signals can be used in

real time to drive motors of active vision systems or to select subregions of images captured from wide field-of-view cameras. The AER communication protocol and the 2D structure of the network make it particularly suitable for processing signals from silicon spiking retinas.

The basic circuits of the chip we present have already been proposed in [11]. The chip we present here comprises improvements in the basic circuits, and additional dynamic components that will be described in Section 3. The chip's improvements over previous implementations arise from the design of new AER interfacing circuits, both for the input decoding stage and the output arbitration, and new synaptic circuits: the Diff-Pair Integrator (DPI) described in [12]. The DPI is a log-domain compact circuit that reproduces the time course of biological post-synaptic currents. Besides having easily and independently tunable gain and time constant, it produces mean currents proportional to the input frequencies, more suitable for the input of the current-mode WTA cell employed as core computational unit in the SAC. This new circuit allows the analysis of the properties of the chip, including the effect of the introduction of additional dynamic properties to the circuits, such as Short-Term Depression (STD) [13, 14] in the input synapses and spike frequency adaptation in the output Integrate and Fire ($I\&F$) neurons [15].

In the next sections we describe the chip's architecture and present experimental results from a two chip system comprising the SAC and a silicon "transient" retina that produces spikes in response to temporal changes in scene contrast.

## 2 The Selective Attention Chip

We fabricated a prototype of the SAC in standard AMS $0.35\mu$m CMOS technology. The chip comprises an array of $32 \times 32$ pixels, each one is $90 \times 45\mu m^2$ and the whole chip with AER digital interface and pads occupies an area of $10mm^2$. The basic functionality of the SAC is to scan the input in order of decreasing activity. The chip input and output signals are asynchronous digital pulses (spikes) that use the *Address Event Representation* (AER) [16]. The input spikes to each pixel are translated into a current (see $I_{ex}$ of Fig.1) by a circuit that models the dynamics of a biological excitatory synapse [12]. A current mode hysteretic Winner–Take–All (WTA) competitive cell compares the input currents of each pixel; the winning cell sources a constant current to the correspondent output leaky Integrate and Fire (I&F) neuron [15]. The spiking neuron in the array then signals which pixel is winning the competition for saliency, and therefore the pixel that receives the highest input frequency. The output spikes of the $I\&F$ neuron are sent also to a feedback inhibitory synapse (see Fig. 1), that subtracts current ($I_{ior}$) from the input node of the WTA cell; the net input current to the winner pixel is then decreased, and a new pixel can eventually be selected. This self-inhibition mechanism is known as Inhibition of Return (IOR) and allows the network to select sequentially the most salient regions of input images, reproducing the attentional scan path.

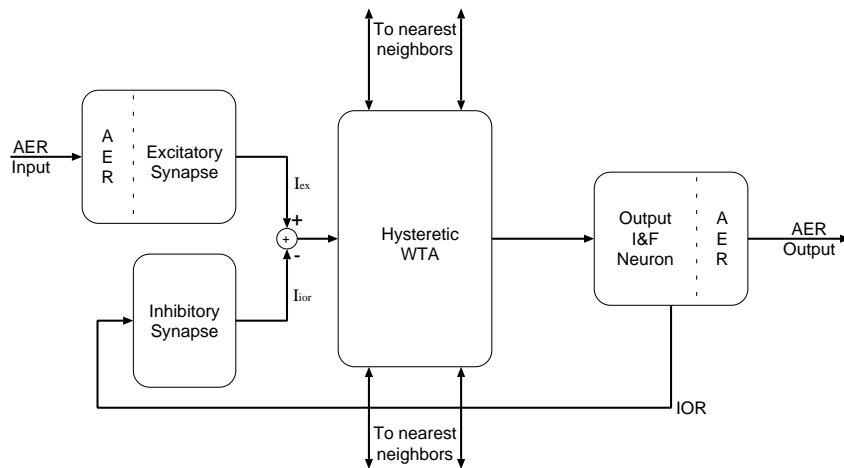

Figure 1: Block diagram of a basic cell of the $32 \times 32$ selective attention architecture.

This basic functionality of the SAC is augmented by the introduction of dynamic properties such as Short-Term Depression (STD) in the input synapses and spike frequency adaptation in the output neuron. STD is a property observed in physiological recordings[17] of synapses that decrease their efficacy when they receive consecutive stimulations. In our synapse the effect of a single spike on the integrated current depends on a voltage, the synaptic weight. The initial weight of the synapse is set by an external voltage reference, then as the synapse receives spikes the effective synaptic weight decreases. STD is a local gain control, that increases sensitivity to changes in the input and makes the synapse insensitive to constant stimulation. Spiking frequency adaptation is another property of neurons that when stimulated with constant input decrease their output firing rate with time. The spiking frequency of the silicon I&F neuron is monotonic with its input current, the adaptation neuron's mechanism decreases the neuron's firing rate with time [15]. We exploit this property to decrease the output bandwidth of the SAC.

The SAC has been designed with tunable parameters that allow to modify the strength of synaptic contributions, the dynamics of synaptic short term depression and of neuronal adaptation, as well as the spatial extent of competition and the dynamics of IOR. All these parameters enrich the dynamics of the network that can be exploited to model the complex selective attention scan path.

## 3 Multi–Chip Selective Attention System

The SAC uses the asynchronous AER SCX (Silicon Cortex) protocol, that allows multiple AER chips to communicate using spikes, just like the cortex, and can be used in multi–chip systems, with multiple senders and multiple receivers [18, 19]. Using this representation the SAC can exchange data, while processing signals in parallel, in real time [20]. The communication protocol used and the SAC's bidimensional architecture make it particularly suitable for processing visual inputs coming from artificial spiking retinas. We built a two chip system, connecting a silicon retina [21] to the SAC input. The retina is an AER asynchronous imager that responds to contrast variations, it has $64 \times 64$ pixels that respond to on and off transients. A dedicated PCI-AER board [18] connects the retina to the SAC, via a look-up table that maps the activity of the $64 \times 64$ pixels of the retina to the $32 \times 32$ pixels of the SAC. In this setup the mapping is linear grouping 4 retina pixels to 1 SAC pixel, more complex mappings, as for example the foveal mapping, will be tested in the future. The board allows also to monitor the activity of both chips on a Linux desktop.

## 4 Experimental Data

We performed preliminary experiments with the two chips setup described in the previous section. We stimulated the retina with two black squares flashing at $6Hz$ on a white background, on a LCD screen, using the matlab PsychoToolbox [22] as shown in Fig. 2. In Fig. 3 we show the response of the two chips to this stimulus: each dot represents the mean firing rate of the correspondent pixel in the chips. The pixels of the retina that focus on the black squares are active and show a high mean firing rate, some other pixels in the array have spontaneous activity. To show the mapping between the retina and the SAC we performed a control experiment: we turned off the competition and the IOR and also we disabled STD and the neuronal adaptation, in this way all the pixels that receive an input activity will be active. All the pixels that receive the input from the pixels of the retina that we stimulate with the black squares are active, more over the spontaneous activity (noise) of the other pixels are "cleaned", thanks to the filtering property of the input synapses. In the next figures we show the response of the system to the stimulus described above, while changing the settings of the SAC. In all the figures the top and bottom boxes show raster plots, respectively of the retina and the SAC: each dot corresponds to a spike emitted by a pixel (or neuron) (y axis) at a certain time (x axis). The middle trace shows the voltage $V_{net}$, that is proportional to the total input current ($I_{ex} - I_{ior}$ of Fig. 1) to the WTA cell that receives input from one of the most active pixels of the retina.

In Fig. 4(a) we show the same data of Fig. 3, the retina sends many spikes every time the black squares appear and disappear from the screen, the WTA input node, with this settings, receives only the excitatory current from the input synapse, as shown by the increase of the voltage $V_{net}$ in correspondence of the retinal spikes. Since in our control experiment there is no competition, all the stimulated pixels are active, as shown in the SAC raster plot. In Fig. 4(b) we show the effect of

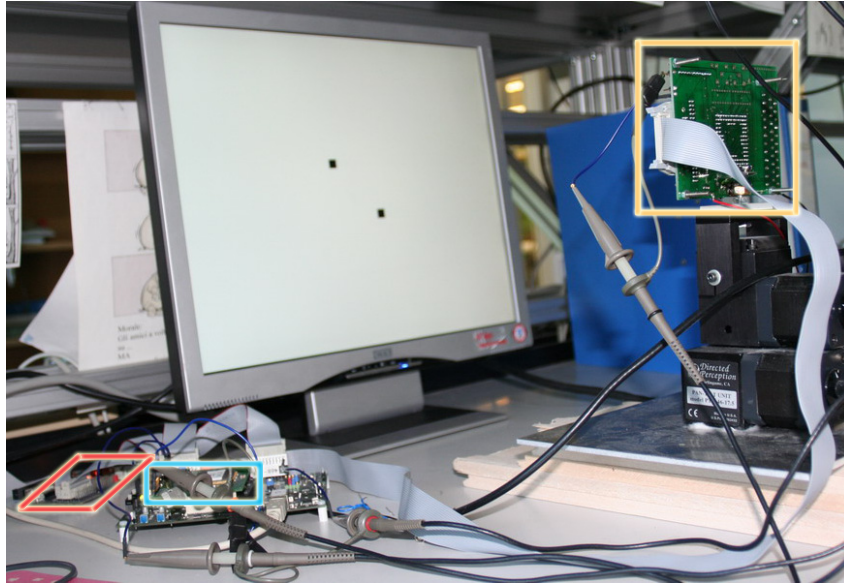

Figure 2: Multi-chip system: The retina (top-right box) is stimulated with an LCD screen, its output is sent to the SAC (bottom-right box) via the PCIAER board (bottom-left box). The activity of the two chips is monitored via the PCIAER board on a Linux desktop.

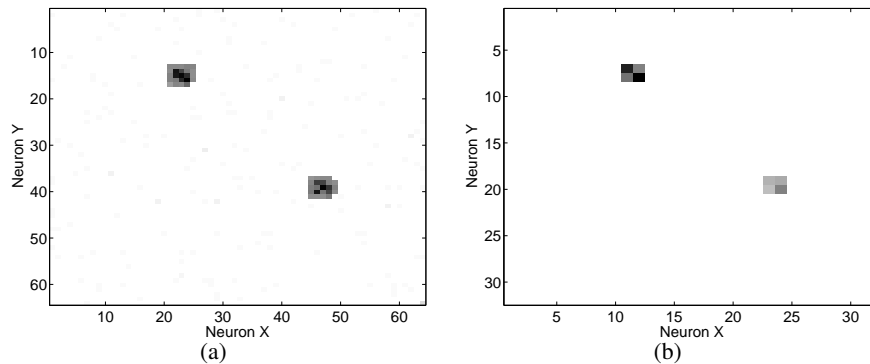

Figure 3: Response of the two chips to an image. (a) The silicon retina is stimulated, via an LCD screen, with two flashing ($6Hz$) black squares on a white background (see Fig. 2). We show the mean firing output of each pixel of the retina. The pixels corresponding to the black squares in the image have higher firing rate than the others, some of the pixels of the retina are spontaneously firing at lower frequencies. (b) The activity of the retina is the input of the SAC: the $64 \times 64$ pixels of the retina are mapped with a ratio $4 : 1$ to the $32 \times 32$ pixels of the SAC. We show the mean firing rate of the SAC pixels in response to the retinal stimulation, when the Winner-Takes-All competition is disabled. In this case the SAC output reflects the input, with some suppression of the noisy pixels due to the filtering properties of the input synaptic circuits.

introducing spike frequency adaptation: in this case the output frequency of each neuron decreases, reducing the output bandwidth and the AER-bus traffic. In Fig. 5 we show the effect of competition and Inhibition of Return. When we turn on the WTA competition only one pixel is selected at any time, therefore only one neuron is firing, as shown in the raster plot of Fig. 5(a); on the node $V_{net}$ we can observe that when the correspondent neuron is winning there is an extra input current, because it doesn't reset to its resting value when the synapse is not active. This positive current implements a form of self-excitation that gives hysteretic properties to the network dynamics, and stabilizes the WTA network. If we turn on the inhibitory synapse (Fig. 5(b)), as soon as the neuron starts to fire,

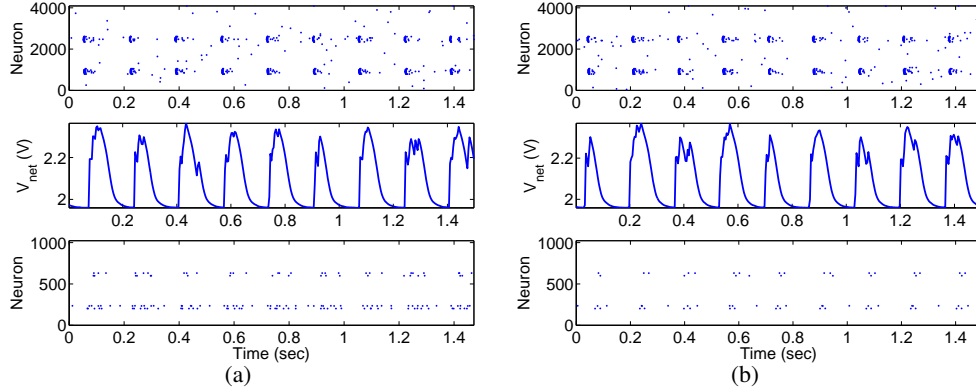

Figure 4: Time response to black squares flashing on a white background: we use the same stimulation and setup described in Fig 3. The top figure shows the raster plot of the retina output, one dot corresponds to a spike produced by one pixel at a specific time. The retina produces events every time the squares appear on or disappear from the screen. The middle plot shows the voltage $V_{net}$ of the input node of the WTA cell correspondent to the synapse that receives input from one of the most active pixel of the retina. The bottom figure shows the raster plot of the SAC neurons. (a) We show the "control" experiment (same as in Fig 3): the competition, IOR, and all the other features of the SAC are turned off, the output of the chip reproduces the input, with some suppression of the pixels that receive very low activity from the retina, thanks to the input synapses filtering properties. In the middle plot $V_{net}$ reflects the effect of the sole input current from the synapse, that integrates the spikes received from the correspondent pixel of the retina. In this case, since the lateral inhibitory connections are switched off, there is no competition and all the output $I\&F$ neurons correspondent to the stimulated input synapses are active. (b) We add spike frequency adaptation to the previous experiment settings, the output firing rate of the neurons is decreased, reducing the bandwidth of the SAC output.

the inhibitory current decreases the total input current to the correspondent WTA cell: the voltage $V_{net}$ reflects this mechanism as it is reset to its resting value even before the input from the retina ceases. The WTA cell is then deselected and the output neuron stops firing, while another neuron is selected and starts firing, as shown in the SAC raster plot. The inhibitory synapse time constant is tunable and when it is slow the effect of inhibition lasts for hundreds of milliseconds after the $I\&F$ stopped firing, in this way we prevent that pixel to be reselected immediately and we can have scan path with many different pixels.

## 5   Conclusions

In this paper we presented a neuromorphic device implementing a Winner–Take–All network comprising dynamic synapses and adaptive neurons. This device is designed to be a part of a multi–chip system for Selective Attention: via an AER communication system it can be interfaced to silicon spiking retinas and to software implementations of associative memories.

We built a multi–chip system with the SAC and a silicon transient retina. The real time measurements allowed by the physical realization of the system are certainly a powerful method to explore the network behavior by changing its parameters. Preliminary experiments confirmed the basic functionality of the SAC and the robustness of the system; the analysis will be extended with the systematic study of STD, IOR, adaptation and lateral excitatory coupling among the nearby cells.

## References

[1]  L. Itti, E. Niebur, and C. Koch. A model of saliency-based visual attention for rapid scene analysis. *IEEE Transactions on Pattern Analysis and Machine Intelligence*, 20(11):1254–1259, 1998.

[2]  H. Bosch, R. Milanese, and A. Labbi. Object segmentation by attention-induced oscillations. In *Proc. IEEE Int. Joint Conf. Neural Networks*, volume 2, pages 1167–1171, 1998.

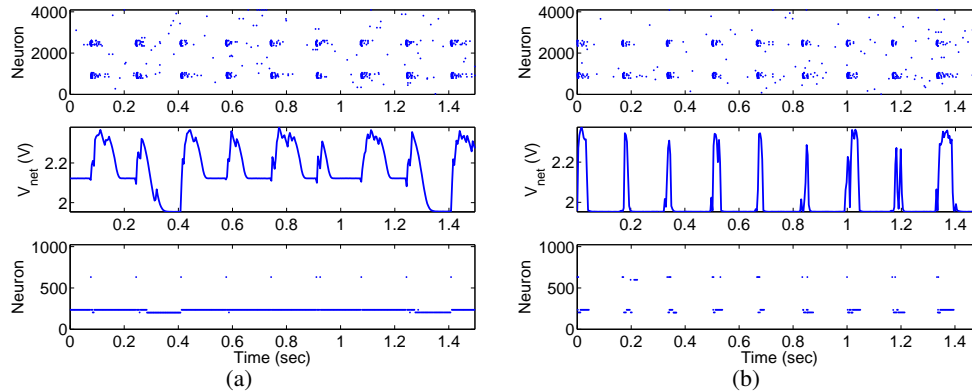

Figure 5: Response of the system with WTA competition and Inhibition of Return. The setup, stimulus and figure content are the same as in Fig. 4. (a) We turn on the WTA competition, and the hysteretic self-excitation. The retina activity is unchanged, the node $V_{net}$ now reflects the input current from the synapse, and also the contribution of the hysteretic current: when the monitored pixel wins the competition for saliency a current is fed in the input node, and $V_{net}$ does not reset to its resting value when the synapse is not active. Now there is only one active neuron in the whole chip, when it does not win the competition for saliency the hysteretic current fades away and another neuron begins spiking. (b) We turn on the inhibitory synapse that implements the self-inhibition (IOR). We can observe the effect of the inhibitory current subtracted from the input node (see text) on $V_{net}$, that with the same input as before sets back to its resting level much faster. The raster plot shows how this mechanism allows to deselect the current winner and select other inputs.

[3] S. Baluja and D.A Pomerleau. Expectation-based selective attention for the visual monitoring and control of a robot vehicle. *Robotics and Autonomous Systems Journal*, 22:329–344, 1997.

[4] V. Brajovic and T. Kanade. Computational sensor for visual tracking with attention. *IEEE Journal of Solid State Circuits*, 33(8):1199–1207, August 1998.

[5] T. K. Horiuchi and C. Koch. Analog VLSI-based modeling of the primate oculomotor system. *Neural Computation*, 11(1):243–265, January 1999.

[6] T. G. Morris, T. K. Horiuchi, and S. P. DeWeerth. Object-based selection within an analog VLSI visual attention system. *IEEE Transactions on Circuits and Systems II*, 45(12):1564–1572, 1998.

[7] G. Indiveri. Modeling selective attention using a neuromorphic analog VLSI device. *Neural Computation*, 12(12):2857–2880, December 2000.

[8] C. Koch and S Ullman. Shifts in selective visual-attention – towards the underlying neural circuitry. *Human Neurobiology*, 4(4):219–227, 1985.

[9] E. Chicca. *A Neuromorphic VLSI System for Modeling Spike–Based Cooperative Competitive Neural Networks*. PhD thesis, ETH Zürich, Zürich, Switzerland, April 2006.

[10] E. Chicca, A. M. Whatley, V. Dante, P. Lichtsteiner, T. Delbruck, P. Del Giudice, R. J. Douglas, and G. Indiveri. A multi-chip pulse-based neuromorphic infrastructure and its application to a model of orientation selectivity. *IEEE Transactions on Circuits and Systems I, Regular Papers*, 2006. (in press).

[11] C. Bartolozzi and G. Indiveri. Selective attention implemented with dynamic synapses and integrate-and-fire neurons. *NeuroComputing, special issue on Brain Inspired Cognitive Systems*, 2005. In press.

[12] C. Bartolozzi and G. Indiveri. Silicon synaptic homeostasis. In *Brain Inspired Cognitive Systems 2006*, 2006.

[13] C. Rasche and R. Hahnloser. Silicon synaptic depression. *Biological Cybernetics*, 84(1):57–62, 2001.

[14] M Boegerhausen, P Suter, and S.-C. Liu. Modeling short-term synaptic depression in silicon. *Neural Computation*, 15(2):331–348, Feb 2003.

[15] G. Indiveri. A low-power adaptive integrate-and-fire neuron circuit. In *Proc. IEEE International Symposium on Circuits and Systems*, pages IV–820–IV–823. IEEE, May 2003.

[16] M. Mahowald. *An Analog VLSI System for Stereoscopic Vision*. Kluwer, Boston, MA, 1994.

[17] L. Abbott, K. Sen, J. Varela, and S. Nelson. Synaptic depression and cortical gain control. *Science*, 275(5297):220–223, 1997.

[18] V. Dante and P. Del Giudice. The PCI-AER interface board. In A. Cohen, R. Douglas, T. Horiuchi, G. Indiveri, C. Koch, T. Sejnowski, and S. Shamma, editors, *2001 Telluride Workshop on Neuromorphic Engineering Report*, pages 99–103, 2001. http://www.ini.unizh.ch/telluride/previous/report01.pdf.

[19] S. R. Deiss, R. J. Douglas, and A. M. Whatley. A pulse-coded communications infrastructure for neuromorphic systems. In W. Maass and C. M. Bishop, editors, *Pulsed Neural Networks*, chapter 6, pages 157–78. MIT Press, 1998.

[20] G. Indiveri. A neuromorphic VLSI device for implementing 2-D selective attention systems. *IEEE Transactions on Neural Networks*, 12(6):1455–1463, November 2001.

[21] P. Lichtsteiner, C. Posch, and T. Delbrück. A 128×128 120dB 30mW asynchronous vision sensor that responds to relative intensity change. In *2006 IEEE ISSCC Digest of Technical Papers*, pages 508–509. IEEE, 2006.

[22] D.H. Brainard. The psychophisics toolbox. *Spatial Vision*, 10:433–436, 1997.

## Acknowledgments

This work was inspired by interactions with the participants of the Neuromorphic Engineering Workshop (http://www.ini.unizh.ch/telluride) and was supported by the EU grant ALAVLSI (IST–2001–38099) and DAISY (FP6-2005-015803), and by the Italian CNR (Centro Nazionale delle Ricerche) fellowship 203.22. The silicon retina was provided by Patrick Lichtsteiner. We further wish to thank Matthias Oster and Dylan Muir for providing AER software tools.
